# All learning is local:
# Multi-agent learning in global reward games

**Yu-Han Chang**
MIT CSAIL
Cambridge, MA 02139
*ychang@csail.mit.edu*

**Tracey Ho**
LIDS, MIT
Cambridge, MA 02139
*trace@mit.edu*

**Leslie Pack Kaelbling**
MIT CSAIL
Cambridge, MA 02139
*lpk@csail.mit.edu*

## Abstract

In large multiagent games, partial observability, coordination, and credit assignment persistently plague attempts to design good learning algorithms. We provide a simple and efficient algorithm that in part uses a linear system to model the world from a single agent's limited perspective, and takes advantage of Kalman filtering to allow an agent to construct a good training signal and learn an effective policy.

## 1   Introduction

Learning in a single-agent stationary-environment setting can be a hard problem, but relative to the multi-agent learning problem, it is easy. The multi-agent learning problem has been approached from a variety of approaches, from game theory to partially observable Markov decision processes. The solutions are often complex. We take a different approach in this paper, presenting a simplifying abstraction and a reward filtering technique that allows computationally efficient and robust learning in large multi-agent environments where other methods may fail or become intractable.

In many multi-agent settings, our learning agent does not have a full view of the world. Other agents may be far away or otherwise obscured. At the very least, our learning agent usually does not have a a complete representation of the internal states of the other agents. This partial observability creates problems when the agent begins to learn about the world, since it cannot see how the other agents are manipulating the environment and thus it cannot ascertain the true world state. It may be appropriate to model the observable world as a non-stationary Markov Decision Process (MDP). A separate problem arises when we train multiple agents using a global reward signal. This is often the case in cooperative games in which all the agents contribute towards attaining some common goal. Even with full observability, the agents would need to overcome a credit assignment problem, since it may be difficult to ascertain which agents were responsible for creating good reward signals. If we cannot even observe what the other agents are doing, how can we begin to reason about their role in obtaining the current reward?

Consider an agent in an MDP, learning to maximize a reward that is a function of its observable state and/or actions. There are many well-studied learning techniques to do this [Sutton and Barto, 1999]. The effects of non-stationarity, partial observability, and global rewards can be thought of as replacing the true reward signal with an alternate signal that is

a non-stationary function of the original reward. Think of the difference between learning with a personal coach and learning in a large class where feedback is given only on collective performance. This causes problems for an agent that is trying to use the collective "global" reward signal to learn an optimal policy. Ideally the agent can recover the original "personal reward signal" and learn using that signal rather than the global reward signal.

We show that in many naturally arising situations of this kind, an effective approach is for an individual agent to model the observed global reward signal as the sum of its own contribution (which is the personal reward signal on which it should base its learning) and a random Markov process (which is the amount of the observed reward due to other agents or external factors). With such a simple model, we can estimate both of these quantities efficiently using an online Kalman filtering process. Many external sources of reward (which could be regarded as noise) can be modeled as or approximated by a random Markov process, so this technique promises broad applicability. This approach is more robust than trying to learn directly from the global reward, allowing agents to learn and converge faster to an optimal or near-optimal policy.

## 2   Related Work

This type of problem has been approached in the past using a variety of techniques. For slowly varying environments, Szita et al. [2002] show that $Q$-learning will converge as long as the variation per time step is small enough. In our case, we attempt to tackle problems where the variation could be larger. Choi et al. [1999] investigate models in which there are "hidden modes". When the environment switches between modes, all the rewards may be altered. This works if we have fairly detailed domain knowledge about the types of modes we expect to encounter. For variation produced by the actions of other agents in the world, or for truly unobservable environmental changes, this technique would not work as well. Auer et al. [1995] show that in arbitrarily varying environments, we can craft a regret-minimizing strategy for playing repeated games. The results are largely theoretical in nature and can yield fairly loose performance bounds, especially in stochastic games. Rather than filtering the rewards as we will do, Ng et al. [1999] show that a potential function can be used to shape the rewards without affecting the learned policy while possibly speeding up convergence. This assumes that learning would converge in the first place, though possibly taking a very long time. Moreover, it requires domain knowledge to craft this shaping function. Wolpert and Tumer [1999] provide a framework called COIN, or collective intelligence, for analyzing distributed reinforcement learning. They stress the importance of choosing utility functions that lead to good policies. Finally, McMahan et al. [2003] discuss learning in the scenario in which the opponent gets to choose the agent's reward function.

The innovative aspect of our approach is to consider the reward signal as merely a signal that is correlated with our true learning signal. We propose a model that captures the relationship between the true reward and the noisy rewards in a wide range of problems. Thus, without assuming much additional domain knowledge, we can use filtering methods to recover the underlying true reward signal from the noisy observed global rewards.

## 3   Mathematical model

The agent assumes that the world possesses one or more unobservable state variables that affect the global reward signal. These unobservable states may include the presence of other agents or changes in the environment. Each agent models the effect of these unobservable state variables on the global reward as an additive noise process $b_t$ that evolves according to $b_{t+1} = b_t + z_t$, where $z_t$ is a zero-mean Gaussian random variable with variance $\sigma_w$.

The global reward that it observes if it is in state $i$ at time $t$ is $g_t = r(i) + b_t$, where $r$ is a vector containing the ideal training rewards $r(i)$ received by the agent at state $i$. The standard model that describes such a linear system is:

$$g_t = Cx_t + v_t, \quad v_t \sim N(0, \Sigma_2)$$
$$x_t = Ax_{t-1} + w_t, \quad w_t \sim N(0, \Sigma_1)$$

In our case, we desire estimates of $x_t = [r_t^T \ \ b_t]^T$. We impart our domain knowledge into the model by specifying the estimated variance and covariance of the components of $x_t$. In our case, we set $\Sigma_2 = 0$ since we assume no observation noise when we experience rewards; $\Sigma_1(j, j) = 0, j \neq |S| + 1$, since the rewards are fixed and do not evolve over time; $\Sigma_1(|S|+1, |S|+1) = \sigma_w$ since the noise term evolves with variance $\sigma_w$. The system matrix is $A = I$, and the observation matrix is $C = [0 \ \ 0 \ldots 1_i \ldots 0 \ \ 0 \ \ 1]$ where the $1_i$ occurs in the $i^{th}$ position when our observed state is state $i$.

Kalman filters [Kalman, 1960] are Bayes optimal, minimum mean-squared-error estimators for linear systems with Gaussian noise. The agent applies the following causal Kalman filtering equations at each time step to obtain maximum likelihood estimates for $b$ and the individual rewards $r(i)$ for each state $i$ given all previous observations. First, the estimate $\hat{x}$ and its covariance matrix $P$ are updated in time based on the linear system model:

$$\hat{x}'_t \ \ = \ \ A\hat{x}_{t-1} \tag{1}$$
$$P'_t \ \ = \ \ AP_{t-1}A^T + \Sigma_1 \tag{2}$$

Then these a priori estimates are updated using the current time period's observation $g_t$:

$$K_t \ \ = \ \ P'_t C^T (CP'_t C^T + \Sigma_2)^{-1} \tag{3}$$
$$\hat{x}_t \ \ = \ \ \hat{x}'_t + K_t(g_t - C\hat{x}'_t) \tag{4}$$
$$P_t \ \ = \ \ (I - K_t C)P'_t \tag{5}$$

As shown, the Kalman filter also gives us the estimation error covariance $P_t$, from which we know the variance of the estimates for $r$ and $b$. We can also compute the likelihood of observing $g_t$ given the model and all the previous observations. This will be handy for evaluating the fit of our model, if needed. We could also create more complicated models if our domain knowledge shows that a different model would be more suitable. For example, if we wanted to capture the effect of an upward bias in the evolution of the noise process (perhaps to model the fact that all the agents are learning and achieving higher rewards), we could add another variable $u$, initialized such that $u_0 > 0$, modifying $x$ to be $x = [r^T \ \ b \ \ u]^T$, and changing our noise term update equation to $b_{t+1} = b_t + u_t + w_t$. In other cases, we might wish to use non-linear models that would require more sophisticated techniques such as extended Kalman filters.

For the learning mechanism, we use a simple tabular $Q$-learning algorithm [Sutton and Barto, 1999], since we wish to focus our attention on the reward signal problem. $Q$-learning keeps a "$Q$-value" for each state-action pair, and proceeds using the following update rule:

$$Q_t(s, a) = (1 - \alpha)Q_{t-1}(s, a) + \alpha(r + \gamma \min_{a'} Q_t(s', a')) \quad , \tag{6}$$

where $0 < \alpha < 1$ is parameter that controls the learning rate, $r$ is the reward signal used for learning at time $t$ given $s$ and $a$, $0 < \gamma \leq 1$ is the discount factor, and $s$, $a$, and $s'$ are the current state, action, and next state of the agent, respectively. Under fairly general conditions, in a stationary MDP, $Q$-learning converges to the optimal policy, expressed as

$$\pi(s) = \mathrm{argmax}_a Q(s, a) \quad .$$

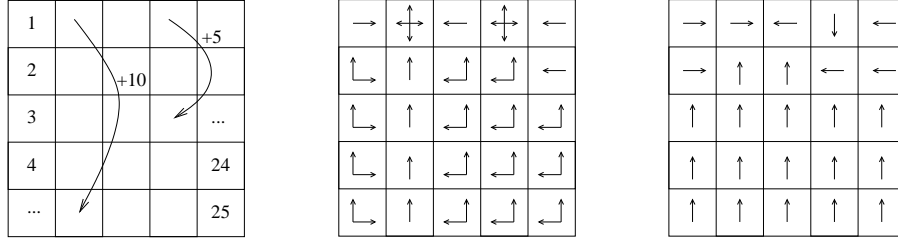

Figure 1: This shows the dynamics of our 5x5 grid world domain. The states correspond to the grid locations, numbered 1,2,3,4,...,24,25. Actions move the agent N,S,E, or W, except in states 6 and 16, where any action takes the agent to state 10 and 18, respectively, shown by the curved arrows in the figure at left. The optimal policy is shown at center, where multiple arrows at one state denotes indifference between the possibilities. A policy learned by our filtering agent is shown at right.

## 4 The filtering learning agent

Like any good student, the filtering learning agent chooses to accept well-deserved praise from its teacher and ignore over-effusive rewards. The good student does not update his behavior at every time step, but only upon observing relevant rewards. The question remains: How does an agent decide upon the relevance of the rewards it sees? We have proposed a model in which undeserved rewards over time are captured by a Markov random process $b$. Using observations from previous states and actions, an agent can approach this question from two perspectives. In the first, each time the agent visits a particular state $i$, it should gain a better sense of the evolution of the random variable $b$ between its last visit and its current visit. It is important to note that rewards are received frequently, thus allowing frequent updating of $b$. Secondly, given an estimate of $b_t$ upon visiting state $i$ at time $t$, it has a better idea of the value of $b_{t+1}$ when it visits state $i'$ at time $t + 1$, since we assume $b_t$ evolves slowly over time. These are the ideas captured by the causal Kalman filter, which only uses the history of past states and observations to provides estimates of $r(i)$ and $b$.

The agent follows this simple algorithm:

1. From initial state $i_0$, take some action $a$, transition to state $i$, and receive reward signal $g_0$. Initialize $\hat{x}_0(i_0) = g_0$ and $\hat{x}_0(|S| + 1) = b_0 = 0$, since $b_0 = 0$.

2. Perform a Kalman update using equations 1-5 to compute the current vector of estimates $\hat{x}$, which includes a component that is the reward estimate $\hat{r}(i_0)$, which will simply equal $g$ this time.

3. From the current state $i$ at time $t$, take another action with some mix of exploration and exploitation; transition to state $j$, receiving reward signal $g_t$. If this is the first visit to state $i$, initialize $\hat{x}_t(i) = g_t - \hat{b}_{t-1}$.

4. Perform a Kalman update using equations 1-5 to compute the current vector of estimates $\hat{x}$, which includes a component that is the reward estimate $\hat{r}(i)$.

5. Update the $Q$-table using $\hat{r}(i)$ in place of $r$ in equation 6; return to Step 3.

The advantage of the Kalman filter is that it requires a constant amount of memory – at no time does it need a full history of states and observations. Instead, it computes a sufficient statistic during each update, $x$ and $P$, which consists of the maximum likelihood estimate of $r$ and $b$, and the covariance matrix of this estimate. Thus, we can run this algorithm online as we learn, and its speed does not deteriorate over time. Its speed is most tied to

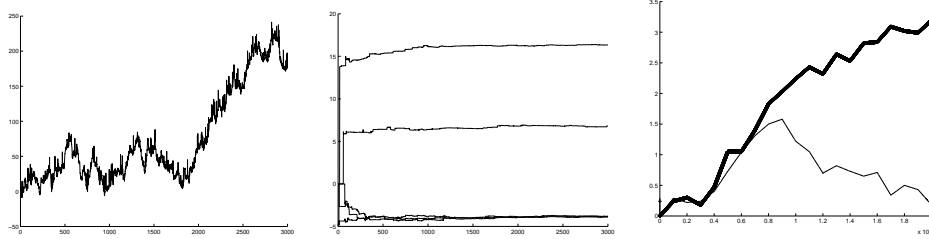

Figure 2: (Left) As the agent is attempting to learn, the reward signal value (y-axis) changes dramatically over time (x-axis) due to the noise term. While the true range of rewards in this grid world domain only falls between 0 and 20, the noisy reward signal ranges from -10 to 250, as shown in the graph at left. (Center) Given this noisy signal, the filtering agent is still able to learn the true underlying rewards, converging to the correct relative values over time, as shown in the middle graph. (Right) The filtering learning agent (bold line) accrues higher rewards over time than the ordinary $Q$-learner (thin line), since it is able to converge to an optimal policy whereas the non-filtering $Q$-learner remains confused.

the number of observation states that we choose to use, since the Kalman update (Eqn. 3) needs to perform a matrix inversion of size $|S| \times |S|$. However, since our model assumes the agent only has access to a limited, local observation space within the true global state space, this computation remains feasible.

## 5    Empirical results

If the world dynamics exactly match the linear model we provide the Kalman filter, then this method will provably converge to the correct reward value estimates and the find the optimal policy under conditions similar to those guaranteeing $Q$-learning's eventual convergence. However, we would rarely expect the world to fit this grossly simplified model. The interesting question concerns situations in which the actual dynamics are clearly different from our model, and whether our filtering agent will still learn a good policy. This section examines the efficacy of the filtering learning agent in several increasingly difficult domains: (1) a single agent domain in which the linear system describes the world perfectly, (2) a single agent domain where the noise is manually adjusted without following the model, (3) a multi-agent setting in which the noise term is meant to encapsulate presence of other agents in the environment, and (4) a more complicated multi-agent setting that simulates an mobile ad-hoc networking domain in which mobile agent nodes try to maximize total network performance.

For ease of exposition, all the domains we use are variants of the popular grid-world domain shown in Figure 1 [Sutton and Barto, 1999]. The agent is able to move North, South, East, or West, and most transitions give the agent zero reward, except all actions from state 6 move the agent directly to state 10 with a reward of 20, and all actions from state 16 move the agent directly to state 18 with a reward of 10. Bumps into the wall cost the agent -1 in reward and move the agent nowhere. We use a discount factor of 0.9.

To demonstrate the basic feasibility of our filtering method, we first create a domain that follows the linear model of the world given in Section 3 perfectly. That is, in each time step, a single agent receives its true reward plus some noise term that evolves as a Markov random process. To achieve this, we simply add a noise term to the grid world domain given in Figure 1. As shown in Figure 2, an agent acting in this domain will receive a large range of reward values due to the evolving noise term. In the example given, sometimes this value ranges as high as 250 even though the maximum reward in the grid world is

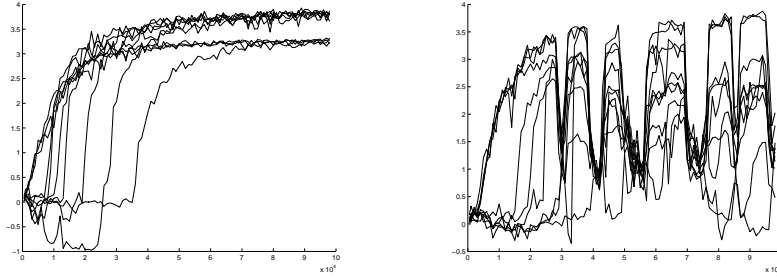

Figure 3: (Left) Filtering agents are able to distinguish their personal rewards from the global reward noise, and thus able to learn optimal policies and maximize their average reward over time in a ten-agent grid-world domain. (Right) In contrast, ordinary $Q$-learning agents do not process the global reward signal and can become confused as the environment changes around them. Graphs show average rewards (y-axis) within 1000-period windows for each of the 10 agents in a typical run of 10000 time periods (x-axis).

20 – the noise term contributes 230 to the reward signal! A standard $Q$-learning agent does not stand a chance at learning anything useful using this reward signal. However, the filtering agent can recover the true reward signal from this noisy signal and use that to learn. Figure 2 shows that the filtering agent can learn the underlying reward signals, converging to these values relatively quickly. The graph to the right compares the performance of the filtering learner to the normal $Q$-learner, showing a clear performance advantage.

The observant reader may note that the learned rewards do not match the true rewards specified by the grid world. Specifically, they are offset by about -4. Instead of mostly 0 rewards at each state, the agent has concluded that most states produce reward of -4. Correspondingly, state 6 now produces a reward of about 16 instead of 20. Since $Q$-learning will still learn the correct optimal policy subject to scaling or translation of the rewards, this is not a problem. This oddity is due to the fact that our model has a degree of freedom in the noise term $b$. Depending on the initial guesses of our algorithm, the estimates for the rewards may be biased. If most of the initial guesses for the rewards underestimated the true reward, then the learned value will be correspondingly lower than the actual true value. In fact, all the learned values will be correspondingly lower by the same amount.

To further test our filtering technique, we next evaluate its performance in a domain that does not conform to our noise model perfectly, but which is still a single agent system. Instead of an external reward term that evolves according to a Gaussian noise process, we adjust the noise manually, introducing positive and negative swings in the reward signal values at arbitrary times. The results are similar to those in the perfectly modeled domain, showing that the filtering method is fairly robust.

The most interesting case occurs when the domain noise is actually caused by other agents learning in the environment. This noise will not evolve according to a Gaussian process, but since the filtering method is fairly robust, we might still expect it to work. If there are enough other agents in the world, then the noise they collectively generate may actually tend towards Gaussian noise. Here we focus on smaller cases where there are 6 or 10 agents operating in the environment. We modify the grid world domain to include multiple simultaneously-acting agents, whose actions do not interfere with each other, but whose reward signal now consists of the sum of all the agents' personal rewards, as given in the basic single agent grid world of Figure 1.

We again compare the performance of the filtering learner to the ordinary $Q$-learning algorithm. As shown in Figure 3, most of the filtering learners quickly converge to the optimal

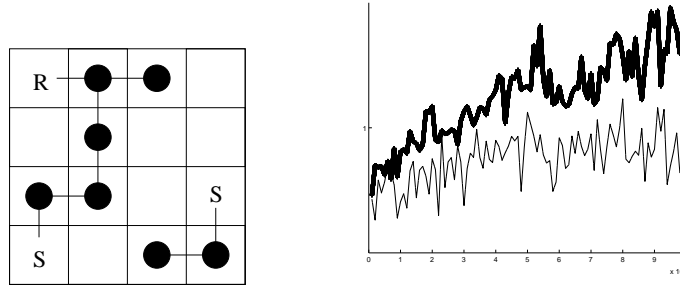

Figure 4: (Left) A snapshot of the 4x4 adhoc-networking domain. S denotes the sources, R is the receiver, and the dots are the learning agents, which act as relay nodes. Lines denote current connections. Note that nodes may overlap. (Right) Graph shows average rewards (y-axis) in 1000-period windows as filtering (bold line) and ordinary (thin line) agents try to learn good policies for acting as network nodes. The filtering agent is able to learn a better policy, resulting in higher network performance (global reward). Graph shows the average for each type of agent over 10 trial runs of 100000 time periods (x-axis) each.

policy. Three of the 10 agents converge to a suboptimal policy that produces slightly lower average rewards. However, this artifact is largely due to our choice of exploration rate, rather than a large error in the estimated reward values. The standard $Q$-learning algorithm also produces decent results at first. Approximately half of the agents find the optimal policy, while the other half are still exploring and learning. An interesting phenomenon occurs when these other agents finally find the optimal policy and begin receiving higher rewards. Suddenly the performance drops drastically for the agents who had found the optimal policy first. Though seemingly strange, this provides a perfect example of the behavior that motivates this paper. When the other agents learn an optimal policy, they begin affecting the global reward, contributing some positive amount rather than a consistent zero. This changes the world dynamics for the agents who had already learned the optimal policy and causes them to "unlearn" their good behavior.

The unstable dynamics of the $Q$-learners could be solved if the agents had full observability, and we could learn using the joint actions of all the agents, as in the work of Claus and Boutilier [1998]. However, since our premise is that agents have only a limited view of the world, the $Q$-learning agents will only exhibit convergence to the optimal policy if they converge to the optimal policy simultaneously. This may take a prohibitively long time, especially as the number of agents grows.

Finally, we apply our filtering method to a more realistic domain. Mobilized ad-hoc networking provides an interesting real-world environment that illustrates the importance of reward filtering due to its high degree of partial observability and a reward signal that depends on the global state. In this domain, there are a number of mobile nodes whose task is to move in such a way as to optimize the connectivity (performance) of the network. Chang et al. [2003] cast this as a reinforcement learning problem. As the nodes move around, connections form between nodes that are within range of one another. These connections allow packets to be transmitted between various sources and receivers scattered among the nodes. The nodes are limited to having only local knowledge of their immediate neighboring grid locations (rather than the numbered state locations as in the original grid world), and thus do not know their absolute location on the grid. They are trained using a global reward signal that is a measure of total network performance, and their actions are limited functions that map their local state to N, S, E, W movements. We also limit their transmission range to a distance of one grid block. For simplicity, the single receiver

is stationary and always occupies the grid location (1,1). Source nodes move around randomly, and in our example here, there are two sources and eight mobile agent nodes in a 4x4 grid. This setup is shown in Figure 4, and the graph shows a comparison of an ordinary $Q$-learner and the filtering learner, plotting the increase in global rewards over time as the agents learn to perform their task as intermediate network nodes. The graph plots average performance over 10 runs, showing the benefit of the filtering process.

## 6    Limitations and extensions

The Kalman filtering framework seems to work well in these example domains. However, there are some cases where we may need to apply more sophisticated techniques. In all the above work, we have assumed that the reward signal is deterministic – each state, action pair only produces a single reward value. There are some domains in which we'd like to model the reward as being stochastic, such as the multi-armed bandit problem. When the stochasticity of the rewards approximates Gaussian noise, we can use the Kalman framework directly. In equation 1, $v$ was set to exhibit zero mean and zero variance. However, allowing some variance would give the model an observation noise term that could reflect the stochasticity of the reward signal.

Finally, in most cases the Kalman filtering method provides a very good estimate of $r$ over time. However, since we cannot guarantee an exact estimate of the reward values when the model is not an exact representation of the world, the agent may make the wrong policy decision sometimes. However, even if the policy is sub-optimal, the error in our derived value function is at least bounded by $\frac{\epsilon}{1-\gamma}$, as long as the $|r(i) - \hat{r}(i)| < \epsilon \quad \forall i$, and $\gamma$ is again the discount rate. In the majority of cases, the estimates are good enough to lead the agent to learning a good policy.

**Conclusion and future work.**    This paper provides the general framework for a new approach to solving large multi-agent problems using a simple model that allows for efficient and robust learning using well-studied tools such as Kalman filtering. As a practical application, we are working on applying these methods to a more realistic version of the mobile ad-hoc networking domain.

## References

[Auer *et al.*, 1995]  P. Auer, N. Cesa-Bianchi, Y. Freund, and R. Schapire.  Gambling in a rigged casino: the adversarial multi-armed bandit problem. In *Proceedings of the 36th Annual Symposium on Foundations of Computer Science*, 1995.

[Chang *et al.*, 2003]  Y. Chang, T. Ho, and L. P. Kaelbling.  Reinforcement learning in mobilized ad-hoc networks. *MIT AI Lab Memo AIM-2003-025*, 2003.

[Choi *et al.*, 1999]  S. Choi, D. Yeung, and N. Zhang. Hidden-mode Markov decision processes. In *IJCAI Workshop on Neural, Symbolic, and Reinforcement Methods for Sequence Learning*, 1999.

[Claus and Boutilier, 1998]  Caroline Claus and Craig Boutilier.  The dynamics of reinforcement learning in cooperative multiaent systems. In *Proceedings of the 15th AAAI*, 1998.

[Kalman, 1960]  R. E. Kalman.  A new approach to linear filtering and prediction problems. *Transactions of the American Society of Mechanical Engineers, Journal of Basic Engineering*, 1960.

[McMahan *et al.*, 2003]  H. McMahan, G. Gordon, and A. Blum.  Planning in the presence of cost functions controlled by an adversary. In *Proceedings of the 20th ICML*, 2003.

[Ng *et al.*, 1999]  Andrew Y. Ng, Daishi Harada, and Stuart Russell. Policy invariance under reward transformations: theory and application to reward shaping. In *Proc. 16th ICML*, 1999.

[Sutton and Barto, 1999]  Richard S. Sutton and Andrew G. Barto.  *Reinforcement Learning: An Introduction*. MIT Press, 1999.

[Szita *et al.*, 2002]  Istvan Szita, Balimt Takacs, and Andras Lorincz.  e-mdps: Learning in varying environments. *Journal of Machine Learning Research*, 2002.

[Wolpert and Tumer, 1999]  D. Wolpert and K. Tumer.  An introduction to collective intelligence. *Tech Report NASA-ARC-IC-99-63*, 1999.
